# Efficient Approaches to Gaussian Process Classification

**Lehel Csató, Ernest Fokoué, Manfred Opper, Bernhard Schottky**
Neural Computing Research Group
School of Engineering and Applied Sciences
Aston University Birmingham B4 7ET, UK.
{opperm,csatol}@aston.ac.uk

**Ole Winther**
Theoretical Physics II, Lund University, Sölvegatan 14 A,
S-223 62 Lund, Sweden
winther@thep.lu.se

## Abstract

We present three simple approximations for the calculation of the posterior mean in Gaussian Process classification. The first two methods are related to mean field ideas known in Statistical Physics. The third approach is based on Bayesian online approach which was motivated by recent results in the Statistical Mechanics of Neural Networks. We present simulation results showing: 1. that the mean field Bayesian evidence may be used for hyperparameter tuning and 2. that the online approach may achieve a low training error fast.

## 1 Introduction

Gaussian processes provide promising non-parametric Bayesian approaches to regression and classification [2, 1]. In these statistical models, it is assumed that the likelihood of an output or target variable $y$ for a given input $\mathbf{x} \in R^N$ can be written as $P(y|a(\mathbf{x}))$ where $a : R^N \to R$ are functions which have a Gaussian prior distribution, i.e. $a$ is (a priori) assumed to be a Gaussian random field. This means that any finite set of field variables $a(\mathbf{x}_i)$, $i = 1, \ldots, l$ are jointly Gaussian distributed with a given covariance $\mathbf{E}[a(\mathbf{x}_i)a(\mathbf{x}_j)] = K(\mathbf{x}_i, \mathbf{x}_j)$ (we will also assume a zero mean throughout the paper).

Predictions on $a(\mathbf{x})$ for novel inputs $\mathbf{x}$, when a set $D$ of $m$ training examples $(\mathbf{x}_i, y_i)$ $i = 1, \ldots, m$, is given, can be computed from the posterior distribution of the $m+1$ variables $a(\mathbf{x})$ and $a(\mathbf{x}_1), \ldots, a(\mathbf{x}_m)$. A major technical problem of the Gaussian process models is the difficulty of computing posterior averages as high dimensional integrals, when the likelihood is not Gaussian. This happens for example in classification problems. So far, a variety of approximation techniques have been discussed: Monte Carlo sampling [2], the MAP approach [4], bounds on the likelihood [3] and a TAP mean field approach [5]. In this paper, we will introduce three different novel methods for approximating the posterior mean of the random field $a(\mathbf{x})$, which we think are simple enough to be used in practical applications. Two of the techniques

are based on mean field ideas from Statistical Mechanics, which in contrast to the previously developed TAP approach are easier to implement. They also yield simple approximations to the total likelihood of the data (the evidence) which can be used to tune the hyperparameters in the covariance kernel $K$ (The Bayesian evidence (or MLII) framework aims at maximizing the likelihood of the data).

We specialize to the case of a binary classification problem, where for simplicity, the class label $y = \pm 1$ is assumed to be noise free and the likelihood is chosen as

$$P(y|a) = \Theta(ya) \ , \tag{1}$$

where $\Theta(x)$ is the unit step function, which equals 1 for $x > 0$ and zero else. We are interested in computing efficient approximations to the posterior mean $\langle a(\mathbf{x}) \rangle$, which we will use for a prediction of the labels via $y = \text{sign}\langle a(\mathbf{x}) \rangle$, where $\langle \ldots \rangle$ denotes the posterior expectation. If the posterior distribution of $a(\mathbf{x})$ is symmetric around its mean, this will give the Bayes optimal prediction.

Before starting, let us add two comments on the likelihood (1). First, the MAP approach (i.e. predicting with the fields $a$ that maximize the posterior) would not be applicable, because it gives the trivial result $a(\mathbf{x}) = 0$. Second, noise can be easily introduced within a *probit* model [2], all subsequent calculations will only be slightly altered. Moreover, the Gaussian average involved in the definition of the probit likelihood can always be shifted from the likelihood into the Gaussian process prior, by a redefinition of the fields $a$ (which does not change the prediction), leaving us with the simple likelihood (1) and a modified process covariance [5].

## 2   Exact Results

At first glance, it may seem that in order to calculate $\langle a(\mathbf{x}) \rangle$ we have to deal with the joint posterior of the fields $a_i = a(\mathbf{x}_i)$, $i = 1, \ldots, m$ together with the field at the test point $a(\mathbf{x})$. This would imply that for any test point, a different new $m + 1$ dimensional average has to be performed. Actually, we will show that this is not the case. As above let $\mathbf{E}$ denote the expectation over the Gaussian prior. The posterior expectation at any point, say $\mathbf{x}$

$$\langle a(\mathbf{x}) \rangle = \frac{\mathbf{E}\left[a(\mathbf{x}) \prod_{j=1}^{m} P(y_j|a_j)\right]}{\mathbf{E}\left[\prod_{j=1}^{m} P(y_j|a_j)\right]} \tag{2}$$

can by integration by parts–for any likelihood–be written as

$$\langle a(\mathbf{x}) \rangle = \sum_j K(\mathbf{x}, \mathbf{x}_j) \alpha_j y_j \quad \text{and} \quad \alpha_j \doteq y_j \left\langle \frac{\partial \ln P(y_j|a_j)}{\partial a_j} \right\rangle \tag{3}$$

showing that $\alpha_j$ is not dependent on the test point $\mathbf{x}$. It is therefore not necessary to compute a $m + 1$ dimensional average for every prediction.

We have chosen the specific definition (3) in order to stress the similarity to predictions with Support Vector Machines (for the likelihood (1), the $\alpha_j$ will come out nonnegative). In the next sections we will develop three approaches for an approximate computation of the $\alpha_j$.

## 3   Mean Field Method I: Ensemble Learning

Our first goal is to approximate the true posterior distribution

$$p(\mathbf{a}|D_m) = \frac{1}{Z} \frac{1}{\sqrt{(2\pi)^m \det \mathbf{K}}} e^{-\frac{1}{2}\mathbf{a}^T \mathbf{K}^{-1} \mathbf{a}} \prod_{j=1}^{m} P(y_j|a_j) \tag{4}$$

of $\mathbf{a} \doteq (a_1, \ldots, a_m)$ by a simpler, tractable distribution $q$. Here, $\mathbf{K}$ denotes the covariance matrix with elements $K_{ij} = K(\mathbf{x}_i, \mathbf{x}_j)$. In the variational mean field approach–known as *ensemble learning* in the Neural Computation Community,– the relative entropy distance $KL(q,p) = \int d\mathbf{a}\, q(\mathbf{a}) \ln \frac{q(\mathbf{a})}{p(\mathbf{a})}$ is minimized in the family of *product distributions* $q(\mathbf{a}) = \prod_{j=1}^{m} q_j(a_j)$. This is in contrast to [3], where a variational bound on the likelihood is computed. We get

$$
KL(q,p) = \sum_i \int da_i q_i(a_i) \ln \frac{q_i(a_i)}{P(y_i|a_i)} +
$$
$$
\frac{1}{2} \sum_{i,j,i \neq j} \left[ \mathbf{K}^{-1} \right]_{ij} \langle a_i \rangle_0 \langle a_j \rangle_0 + \frac{1}{2} \sum_i \left[ \mathbf{K}^{-1} \right]_{ii} \langle a_i^2 \rangle_0
$$

where $\langle \ldots \rangle_0$ denotes expectation w.r.t. $q$. By setting the functional derivative of $KL(q,p)$ with respect to $q_i(a)$ equal to zero, we find that the best product distribution is a Gaussian prior times the original Likelihood:

$$
q_i(a) \propto P(y_i|a) \frac{1}{\sqrt{2\pi\lambda_i}} e^{-\frac{(a-m_i)^2}{2\lambda_i}} , \tag{5}
$$

where $m_i = -\lambda_i \sum_{j,j \neq i} (\mathbf{K}^{-1})_{ij} \langle a_j \rangle_0$ and $\lambda_i = \left[ \mathbf{K}^{-1} \right]_{ii}^{-1}$. Using this specific form for the approximated posterior $q(a)$, replacing the average over the true posterior in (3) by the approximation (5), we get (using the likelihood (1)) a set of $m$ nonlinear equations in the unknowns $\alpha_j$:

$$
\alpha_j = y_j \left\langle \frac{\partial \ln P(y_j|a_j)}{\partial a_j} \right\rangle_0 = \frac{1}{\sqrt{\lambda_j}} \frac{D(\frac{m_j}{\sqrt{\lambda_j}})}{\Phi(y_j \frac{m_j}{\sqrt{\lambda_j}})} \quad \text{and} \quad m_j = \sum_i K_{ji} y_i \alpha_i - \lambda_j y_j \alpha_j ,
$$
$$
\tag{6}
$$

where $D(z) = e^{-z^2/2}/\sqrt{2\pi}$ and $\Phi(z) = \int_{-\infty}^{z} dt\, D(t)$. As a useful byproduct of the variational approximation, an upper bound on the Bayesian evidence $P(D) = \int d\mathbf{a}\, \pi(\mathbf{a})P(D|\mathbf{a})$ can be derived. ($\pi$ denotes the Gaussian process prior and $P(D|\mathbf{a}) = \prod_{j=1}^{m} P(y_j|a_j)$). The bound can be written in terms of the mean field 'free energy' as

$$
-\ln P(D) \leq \mathbf{E}_q \ln q(\mathbf{a}) - \mathbf{E}_q \ln[\pi(\mathbf{a})P(D|\mathbf{a})]
$$
$$
= -\sum_i \ln \Phi \left( y_i \frac{m_j}{\sqrt{\lambda_i}} \right) + \frac{1}{2} \sum_{ij} y_i \alpha_i (K_{ij} - \delta_{ij}\lambda_i) y_j \alpha_j \tag{7}
$$
$$
+ \frac{1}{2} \ln \det \mathbf{K} - \frac{1}{2} \sum_i \ln \lambda_i
$$

which can be used as a yardstick for selecting appropriate hyperparameters in the covariance kernel.

The ensemble learning approach has the little drawback, that it requires inversion of the covariance matrix $\mathbf{K}$ and, for the free energy (7) one must compute a determinant. A second, simpler approximation avoids these computations.

## 4  Mean Field Theory II: A 'Naive' Approach

The second mean field theory aims at working directly with the variables $\alpha_j$. As a starting point, we consider the partition function (evidence),

$$
Z = P(D) = \int d\mathbf{z}\, e^{-\frac{1}{2}\mathbf{z}^T \mathbf{K} \mathbf{z}} \prod_{j=1}^{m} \hat{P}(y_j|z_j) , \tag{8}
$$

which follows from (4) by a standard Gaussian integration, introducing the Fourier transform of the Likelihood $\hat{P}(y|z) = \int \frac{da}{2\pi} e^{iaz} P(y|a)$ with $i$ being the imaginary unit. It is tempting to view (8) as a normalizing partition function for a Gaussian process $z_i$ having covariance matrix $\mathbf{K}^{-1}$ and likelihood $\hat{P}$. Unfortunately, $\hat{P}$ is not a real number and precludes a proper probabilistic interpretation. Neverthe­less, dealing formally with the complex measure defined by (8), integration by parts shows that one has $y_j \alpha_j = -i\langle z_j \rangle_*$, where the brackets $\langle \ldots \rangle_*$ denote a average over the complex measure. This suggests a simple approximation for calculating the $\alpha_j$. One may think of trying a saddle-point (or steepest descent) approximation to (8) and replace $\langle z_j \rangle_*$ by the value of $z_j$ (in the complex $z$ plane) which makes the integrand stationary thereby neglecting the fluctuations of the $z_j$. Hence, this approximation would treat expectations of products as $\langle z_i z_j \rangle_*$ as $\langle z_i \rangle_* \langle z_j \rangle_*$, which may be reasonable for $i \neq j$, but definitely not for the self-correlation $i = j$. Ac­cording to the general formalism of mean field theories (outlined e.g. in [6]), one can improve on that idea, by treating the 'self-interactions' $z_i^2$ separately. This can be done by replacing all $z_i$ (except in the form $z_i^2$) by a new variable $\mu_i$ by inserting a Dirac $\delta$ function representation $\delta(z - \mu) = \int \frac{dm}{2\pi} e^{-im(z-\mu)}$ into (8) and integrate over the $z$ and $a$ variables exactly (the integral factorizes), and finally perform a saddle-point integration over the $m$ and $\mu$ variables. The details of this calculation will be given elsewhere. Within the saddle-point approximation, we get the system of nonlinear equations

$$\alpha_j = -iy_j\mu_j = \frac{1}{\sqrt{K_{jj}}} \frac{D(\frac{m_j}{\sqrt{K_{jj}}})}{\Phi(y_j\frac{m_j}{\sqrt{K_{jj}}})} \quad \text{and} \quad m_j = \sum_{i,i\neq j} K_{ji}(-i\mu_i) = \sum_{i,i\neq j} K_{ji}y_i\alpha_i \quad (9)$$

which is of the same form as (6) with $\lambda_j$ replaced by the simpler $K_{jj}$. These equations have also been derived by us in [5] using a Callen identity, but our present derivation allows also for an approximation to the evidence. By plugging the saddle-point values back into the partition function, we get

$$-\ln P(D) \approx -\sum_i \ln \Phi\left(y_i\frac{m_i}{\sqrt{K_{ii}}}\right) + \frac{1}{2}\sum_{ij} y_i\alpha_i(K_{ij} - \delta_{ij}K_{ii})y_j\alpha_j$$

which is also simpler to compute than (7) but does not give a bound on the true evidence.

## 5  A sequential Approach

Both previous algorithms do not give an explicit expression for the posterior mean, but require the solution of a set of nonlinear equations. These must be obtained by an iterative procedure. We now present a different approach for an approximate computation of the posterior mean, which is based on a single sequential sweep through the whole dataset giving an explicit update of the posterior.

The algorithm is based on a recently proposed Bayesian approach to online learning (see [8] and the articles of Opper and Winther& Solla in [9]). Its basic idea applied to the Gaussian process scenario, is as follows: Suppose, that $q_t$ is a Gaussian approximation to the posterior after having seen $t$ examples. This means that we approximate the posterior process by a Gaussian process with mean $\langle a(\mathbf{x})\rangle_t$ and covariance $K_t(\mathbf{x},\mathbf{y})$, starting with $\langle a(\mathbf{x})\rangle_0 = 0$ and $K_0(\mathbf{x},\mathbf{y}) = K(\mathbf{x},\mathbf{y})$. After a new data point $y_{t+1}$ is observed, the posterior is updated according to Bayes rule. The new non-Gaussian posterior $\hat{q}_t$ is projected back into the family of Gaussians by choosing the closest Gaussian $q_{t+1}$ minimizing the relative entropy $KL(\hat{q}_t, q_{t+1})$

in order to keep the loss of information small. This projection is equivalent to a matching of the first two moments of $\hat{q}_t$ and $q_{t+1}$. E.g., for the first moment we get

$$\langle a(\mathbf{x})\rangle_{t+1} = \frac{\langle a(\mathbf{x})\ P(y_{t+1}|a(\mathbf{x}_{t+1}))\rangle_t}{\langle P(y_{t+1}|a(\mathbf{x}_{t+1}))\rangle_t} = \langle a(\mathbf{x})\rangle_t + \kappa_1(t)K_t(\mathbf{x},\mathbf{x}_{t+1})$$

where the second line follows again from an integration by parts and $\kappa_1(t) = \frac{y_{t+1}}{\sigma}\frac{\phi'(z_t)}{\phi(z_t)}$ with $z_t = \frac{y_{t+1}\langle a(\mathbf{x}_{t+1})\rangle_t}{\sigma(t)}$ and $\sigma^2(t) = K_t(\mathbf{x}_{t+1},\mathbf{x}_{t+1})$. This recursion and the corresponding one for $K_t$ can be solved by the ansatz

$$\langle a(\mathbf{x})\rangle_t \quad = \quad \sum_{j=1}^{t} K(\mathbf{x},\mathbf{x}_j)y_j\alpha_j(t) \tag{10}$$

$$K_t(\mathbf{x},\mathbf{y}) \quad = \quad \sum_{i,j} K(\mathbf{x},\mathbf{x}_i)C_{ij}(t)K(\mathbf{x},\mathbf{x}_j) + K(\mathbf{x},\mathbf{y}) \tag{11}$$

where the vector $\alpha(t) = (a_1,\ldots,a_t,0,0,\ldots)$ and the matrix $C(t)$ (which has also only $t \times t$ nonzero elements) are updated as

$$\alpha(t+1) \quad = \quad \alpha(t) + \kappa_1(t)\left(\mathbf{C}(t)\mathbf{k_{t+1}} + \mathbf{e_{t+1}}\right)\otimes\mathbf{y}$$

$$\mathbf{C}(t+1) \quad = \quad \mathbf{C}(t) + \kappa_2(t)\left(\mathbf{C}(t)\mathbf{k_{t+1}} + \mathbf{e_{t+1}}\right)\left(\mathbf{C}(t)\mathbf{k_{t+1}} + \mathbf{e_{t+1}}\right)^T \tag{12}$$

where $\kappa_2(t) = \frac{1}{\sigma^2}\left\{\frac{\Phi''(z_t)}{\Phi(z_t)} - \left(\frac{\Phi'(z_t)}{\Phi(z_t)}\right)^2\right\}$, $\mathbf{k_t}$ is the vector with elements $K_{tj}$, $j = 1\ldots,t$ and $\otimes$ denotes the element-wise product between vectors. The sequential algorithm defined by (10)-(12) has the advantage of not requiring any matrix inversions. There is also no need to solve a numerical optimization problem at each time as in the approach of [11] where a different update of a Gaussian posterior approximation was proposed. Since we do not require a linearization of the likelihood, the method is not equivalent to the extended Kalman Filter approach.

Since it is possible to compute the evidence of the new datapoint $P(y_{t+1}) = \langle P(y_{t+1}|a_{t+1})\rangle_t$ based on the old posterior, we can compute a further approximation to the log evidence for $m$ data via $\ln P(D_m) = \sum_{t=1}^{m-1}\ln\langle P(y_{t+1}|a_{t+1})\rangle_t$.

## 6  Simulations

We present two sets of simulations for the mean field approaches. In the first, we test the Bayesian evidence framework for tuning the hyperparameters of the covariance function (kernel). In the second, we test the ability of the sequential approach to achieve low training error and a stable test error for fixed hyperparameters.

For the evidence framework, we give simulation results for both mean field free energies (7) and (10) on a single data set, 'Pima Indian Diabetes (with 200/332 training/test-examples and input dimensionality $d = 7$) [7]. The results should therefore not be taken as a conclusive evidence for the merits of these approaches, but simply as an indication that they may give reasonable results. We use the radial basis function covariance function $K(\mathbf{x},\mathbf{x}') = \exp\left(-\frac{1}{2}\sum_l^d w_l(x_l - x_l')^2\right)$. A diagonal term $v$ is added to the covariance matrix corresponding to a Gaussian noise added to the fields with variance $v$ [5]. The free energy, $-\ln P(D)$ is minimized by gradient descent with respect to $v$ and the lengthscale parameters $w_1,\ldots,w_d$ and the mean field equations for $\alpha_j$ are solved by iteration before each update of the hyperparameters (further details will be given elsewhere). Figure 1 shows the evolution of the naive mean free energy and the test error starting from uniform

ws. It typically requires of the order of 10 iteration steps of the $\alpha_j$-equations between each hyperparameter update. We also used hybrid approaches, where the free energy was minimized by one mean field algorithm and the hyperparameters used in the other. As it may be seen from table 1, the naive mean field theory can overestimate the free energy (since the ensemble free energy is an upper bound to the free energy). The overestimation is not nearly as severe at the minimum of the naive mean field free energy. Another interesting observation is that as long as the same hyperparameters are used the actual performance (as measured by the test error) is not very sensitive to the algorithm used. This also seems to be the case for the TAP mean field approach and Support Vector Machines [5].

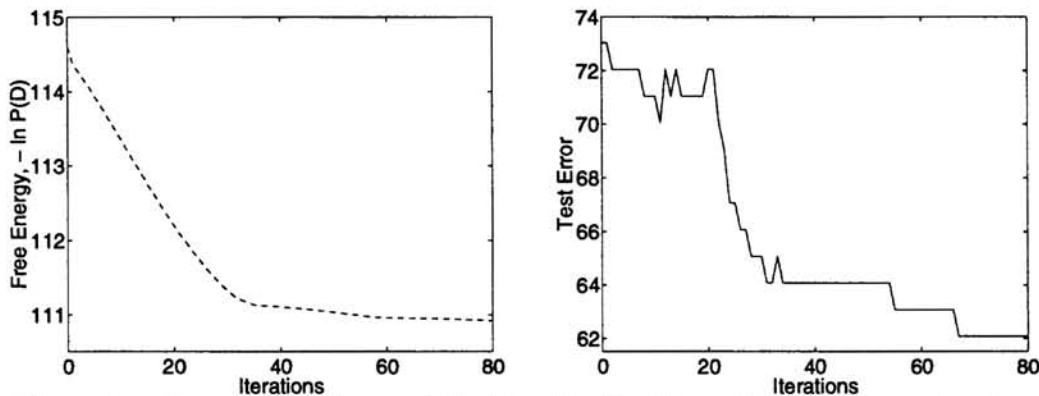

Figure 1: Hyperparameter optimization for the Pima Indians data set using the naive mean field free energy. Left figure: The free energy as a function of the number of hyperparameter updates. Right figure: The test error count (out of 332) as a function of the number of hyperparameter updates.

Table 1: Pima Indians dataset. Hyperparameters found by free energy minimization. Left column gives the free energy $-\ln P(D)$ used in hyperparameter optimization. Test error counts in range 63- 75 have previously been reported [5]

|  | Ensemble MF | | Naive MF | |
| --- | --- | --- | --- | --- |
| Free Energy minimization | Error | $-\ln P(D)$ | Error | $-\ln P(D)$ |
| Ensemble Mean Field, eq. (7) | 72 | 100.6 | 70 | 183.2 |
| Naive Mean Field, eq. (10) | 62 | 107.0 | 62 | 110.9 |

For the sequential algorithm, we have studied the *sonar* [10] and *crab* [7] datasets. Since we have not computed an approximation to the evidence so far, a simple fixed polynomial kernel was used. Although a probabilistic justification of the algorithm is only valid, when a *single* sweep through the data is used (the independence of the data is assumed), it is tempting to reuse the same data and iterate the procedure as a heuristic. The two plots show that in this way, only a small improvement is obtained, and it seems that the method is rather efficient in extracting the information from the data in a single presentation. For the sonar dataset, a single sweep is enough to achieve zero training error.

**Acknowledgements:** BS would like to thank the Leverhulme Trust for their support (F/250/K). The work was also supported by EPSRC Grant GR/L52093.

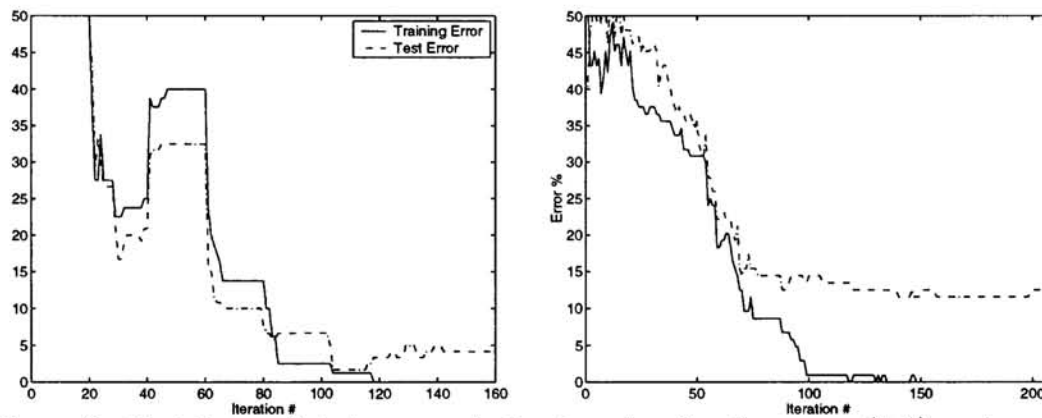

Figure 2: Training and test errors during learning for the sonar (left) and crab dataset (right). The vertical dash-dotted line marks the end of the training set and the starting point of reusing of it. The kernel function used is $K(\mathbf{x}, \mathbf{x}') = (1 + \mathbf{x} \cdot \mathbf{x}'/m)^k$ with order $k = 2$ ($m$ is the dimension of inputs).

# References

[1] Williams C.K.I. and Rasmussen C.E., Gaussian Processes for Regression, in *Neural Information Processing Systems 8*, Touretzky D.S, Mozer M.C. and Hasselmo M.E. (eds.), 514-520, MIT Press (1996).

[2] Neal R.M, *Monte Carlo Implementation of Gaussian Process Models for Bayesian Regression and Classification*, Technical Report 9702, Department of Statistics, University of Toronto (1997).

[3] Gibbs M.N. and Mackay D.J.C., *Variational Gaussian Process Classifiers*, Preprint Cambridge University (1997).

[4] Williams C.K.I. and Barber D, *Bayesian Classification with Gaussian Processes*, IEEE Trans Pattern Analysis and Machine Intelligence, **20** 1342-1351 (1998).

[5] Opper M. and Winther O. *Gaussian Processes for Classification: Mean Field Algorithms*, Submitted to Neural Computation, http://www.thep.lu.se /tf2/staff/winther/ (1999).

[6] Zinn-Justin J, *Quantum Field Theory and Critical Phenomena*, Clarendon Press Oxford (1990).

[7] Ripley B.D, *Pattern Recognition and Neural Networks*, Cambridge University Press (1996).

[8] Opper M., Online versus Offline Learning from Random Examples: General Results, Phys. Rev. Lett. 77, 4671 (1996).

[9] *Online Learning in Neural Networks*, Cambridge University Press, D. Saad (ed.) (1998).

[10] Gorman R.P and Sejnowski T.J, *Analysis of hidden units in a layered network trained to classify sonar targets*, Neural Networks 1, (1988).

[11] Jaakkola T. and Haussler D. *Probabilistic kernel regression*, In Online Proceedings of 7-th Int. Workshop on AI and Statistics (1999), http://uncertainty99.microsoft.com/proceedings.htm.
